# Predicting human gaze using low-level saliency combined with face detection

**Moran Cerf**
Computation and Neural Systems
California Institute of Technology
Pasadena, CA 91125
moran@klab.caltech.edu

**Jonathan Harel**
Electrical Engineering
California Institute of Technology
Pasadena, CA 91125
harel@klab.caltech.edu

**Wolfgang Einhäuser**
Institute of Computational Science
Swiss Federal Institute of Technology (ETH)
Zurich, Switzerland
wolfgang.einhaeuser@inf.ethz.ch

**Christof Koch**
Computation and Neural Systems
California Institute of Technology
Pasadena, CA 91125
koch@klab.caltech.edu

## Abstract

Under natural viewing conditions, human observers shift their gaze to allocate processing resources to subsets of the visual input. Many computational models try to predict such voluntary eye and attentional shifts. Although the important role of high level stimulus properties (e.g., semantic information) in search stands undisputed, most models are based on low-level image properties. We here demonstrate that a combined model of face detection and low-level saliency significantly outperforms a low-level model in predicting locations humans fixate on, based on eye-movement recordings of humans observing photographs of natural scenes, most of which contained at least one person. Observers, even when not instructed to look for anything particular, fixate on a face with a probability of over 80% within their first two fixations; furthermore, they exhibit more similar scanpaths when faces are present. Remarkably, our model's predictive performance in images that do *not* contain faces is not impaired, and is even improved in some cases by spurious face detector responses.

## 1   Introduction

Although understanding attention is interesting purely from a scientific perspective, there are numerous applications in engineering, marketing and even art that can benefit from the understanding of both attention *per se*, and the allocation of resources for attention and eye movements. One accessible correlate of human attention is the fixation pattern in scanpaths [1], which has long been of interest to the vision community [2]. Commonalities between different individuals' fixation patterns allow computational models to predict where people look, and in which order [3]. There are several models for predicting observers' fixations [4], some of which are inspired by putative neural mechanisms. A frequently referenced model for fixation prediction is the Itti *et al.* saliency map model (SM) [5]. This "bottom-up" approach is based on contrasts of intrinsic images features such as color, orientation, intensity, flicker, motion and so on, without any explicit information about higher order scene structure, semantics, context or task-related ("top-down") factors, which may be crucial for attentional allocation [6]. Such a bottom-up saliency model works well when higher order semantics are reflected in low-level features (as is often the case for isolated objects, and even for reasonably cluttered scenes), but tends to fail if other factors dominate: e.g., in search tasks [7, 8], strong contextual effects [9], or in free-viewing of images without clearly isolated objects, such as

forest scenes or foliage [10]. Here, we test how images containing faces - ecologically highly relevant objects - influence variability of scanpaths across subjects. In a second step, we improve the standard saliency model by adding a "face channel" based on an established face detector algorithm. Although there is an ongoing debate regarding the exact mechanisms which underlie face detection, there is no argument that a normal subject (in contrast to autistic patients) will not interpret a face purely as a reddish blob with four lines, but as a much more significant entity ([11, 12]. In fact, there is mounting evidence of infants' preference for face-like patterns before they can even consciously perceive the category of faces [13], which is crucial for emotion and social processing ([13, 14, 15, 16]).

Face detection is a well investigated area of machine vision. There are numerous computer-vision models for face detection with good results ([17, 18, 19, 20]). One widely used model for face recognition is the Viola & Jones [21] feature-based template matching algorithm (VJ). There have been previous attempts to incorporate face detection into a saliency model. However, they have either relied on biasing a color channel toward skin hue [22] - and thus being ineffective in many cases nor being face-selective *per se* - or they have suffered from lack of generality [23]. We here propose a system which combines the bottom-up saliency map model of Itti *et al.* [5] with the Viola & Jones face detector.

The contributions of this study are: (1) Experimental data showing that subjects exhibit significantly less variable scanpaths when viewing natural images containing faces, marked by a strong tendency to fixate on faces early. (2) A novel saliency model which combines a face detector with intensity, color, and orientation information. (3) Quantitative results on two versions of this saliency model, including one extended from a recent graph-based approach, which show that, compared to previous approaches, it better predicts subjects' fixations on images with faces, and predicts as well otherwise.

## 2 Methods

### 2.1 Experimental procedures

Seven subjects viewed a set of 250 images ($1024 \times 768$ pixels) in a three phase experiment. 200 of the images included frontal faces of various people; 50 images contained no faces but were otherwise identical, allowing a comparison of viewing a particular scene with and without a face. In the first ("free-viewing") phase of the experiment, 200 of these images (the same subset for each subject) were presented to subjects for 2 s, after which they were instructed to answer "How interesting was the image?" using a scale of 1-9 (9 being the most interesting). Subjects were not instructed to look at anything in particular; their only task was to rate the entire image. In the second ("search") phase, subjects viewed another 200 image subset in the same setup, only this time they were initially presented with a probe image (either a face, or an object in the scene: banana, cell phone, toy car, etc.) for 600 ms after which one of the 200 images appeared for 2 s. They were then asked to indicate whether that image contained the probe. Half of the trials had the target probe present. In half of those the probe was a face. Early studies suggest that there should be a difference between free-viewing of a scene, and task-dependent viewing of it [2, 4, 6, 7, 24]. We used the second task to test if there are any differences in the fixation orders and viewing patterns between free-viewing and task-dependent viewing of images with faces. In the third phase, subjects performed a 100 images recognition memory task where they had to answer with y/n whether they had seen the image before. 50 of the images were taken from the experimental set and 50 were new. Subjects' mean performance was 97.5% correct, verifying that they were indeed alert during the experiment.

The images were introduced as "regular images that one can expect to find in an everyday personal photo album". Scenes were indoors and outdoors still images (see examples in Fig. 1). Images included faces in various skin colors, age groups, and positions (no image had the face at the center as this was the starting fixation location in all trials). A few images had face-like objects (see balloon in Fig. 1, panel 3), animal faces, and objects that had irregular faces in them (masks, the Egyptian sphinx face, etc.). Faces also vary in size (percentage of the entire image). The average face was $5\% \pm 1\%$ (mean $\pm$ s.d.) of the entire image - between $1°$ to $5°$ of the visual field; we also varied the number of faces in the image between 1-6, with a mean of $1.1 \pm 0.48$. Image order was randomized throughout, and subjects were naïve to the purpose of the experiment. Subjects fixated on a cross in the center before each image onset. Eye-position data were acquired at 1000 Hz using an Eyelink 1000 (SR Research, Osgoode, Canada) eye-tracking device. The images were presented on a CRT

screen (120 Hz), using Matlab's Psychophysics and eyelink toolbox extensions ([25, 26]). Stimulus luminance was linear in pixel values. The distance between the screen and the subject was 80 cm, giving a total visual angle for each image of $28° \times 21°$. Subjects used a chin-rest to stabilize their head. Data were acquired from the right eye alone. All subjects had uncorrected normal eyesight.

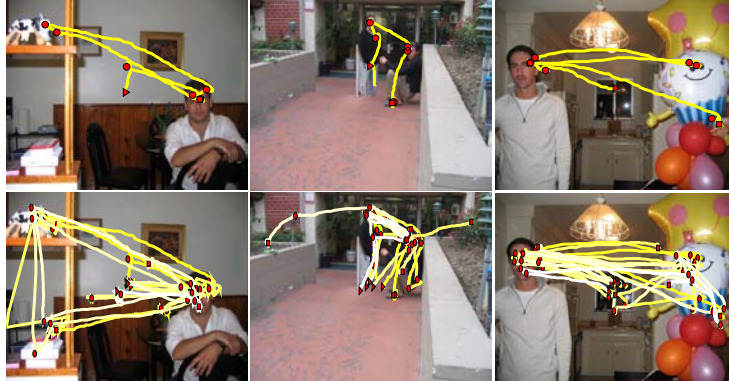

Figure 1: Examples of stimuli during the "free-viewinng" phase. Notice that faces have neutral expressions. Upper 3 panels include scanpaths of one individual. The red triangle marks the first and the red square the last fixation, the yellow line the scanpath, and the red circles the subsequent fixations. Lower panels show scanpaths of all 7 subjects. The trend of visiting the faces first - typically within the $1^{st}$ or $2^{nd}$ fixation - is evident. All images are available at *http://www.klab.caltech.edu/~moran/db/faces/*.

## 2.2 Combining face detection with various saliency algorithms

We tried to predict the attentional allocation via fixation patterns of the subjects using various saliency maps. In particular, we computed four different saliency maps for each of the images in our data set: (1) a saliency map based on the model of [5] (SM), (2) a graph-based saliency map according to [27] (GBSM), (3) a map which combines SM with face-detection via VJ (SM+VJ), and (4) a saliency map combining the outputs of GBSM and VJ (GBSM+VJ). Each saliency map was represented as a positive valued heat map over the image plane.

SM is based on computing feature maps, followed by center-surround operations which highlight local gradients, followed by a normalization step prior to combining the feature channels. We used the "Maxnorm" normalization scheme which is a spatial competition mechanism based on the squared ratio of global maximum over average local maximum. This promotes feature maps with one conspicuous location to the detriment of maps presenting numerous conspicuous locations. The graph-based saliency map model (GBSM) employs spectral techniques in lieu of center surround subtraction and "Maxnorm" normalization, using only local computations. GBSM has shown more robust correlation with human fixation data compared with standard SM [27].

For face detection, we used the *Intel Open Source Computer Vision Library* ("OpenCV") [28] implementation of [21]. This implementation rapidly processes images while achieving high detection rates. An efficient classifier built using the Ada-Boost learning algorithm is used to select a small number of critical visual features from a large set of potential candidates. Combining classifiers in a cascade allows background regions of the image to be quickly discarded, so that more cycles process promising face-like regions using a template matching scheme. The detection is done by applying a classifier to a sliding search window of 24x24 pixels. The detectors are made of three joined black and white rectangles, either up-right or rotated by $45°$. The values at each point are calculated as a weighted sum of two components: the pixel sum over the black rectangles and the sum over the whole detector area. The classifiers are combined to make a boosted cascade with classifiers going from simple to more complex, each possibly rejecting the candidate window as "not a face" [28]. This implementation of the *facedetect* module was used with the standard default training set of the original model. We used it to form a "Faces conspicuity map", or "Face channel"

by convolving delta functions at the (x,y) detected facial centers with 2D Gaussians having standard deviation equal to estimated facial radius. The values of this map were normalized to a fixed range.

For both SM and GBSM, we computed the combined saliency map as the mean of the normalized color (C), orientation (O), and intensity (I) maps [5]:

$$\frac{1}{3}(N(\overline{I}) + N(\overline{C}) + N(\overline{O}))$$

And for SM+VJ and GBSM+VJ, we incorporated the normalized face conspicuity map (F) into this mean (see Fig 2):

$$\frac{1}{4}(N(\overline{I}) + N(\overline{C}) + N(\overline{O}) + N(\overline{F}))$$

This is our combined face detector/saliency model. Although we could have explored the space of combinations which would optimize predictive performance, we chose to use this simplest possible combination, since it is the least complicated to analyze, and also provides us with first intuition for further studies.

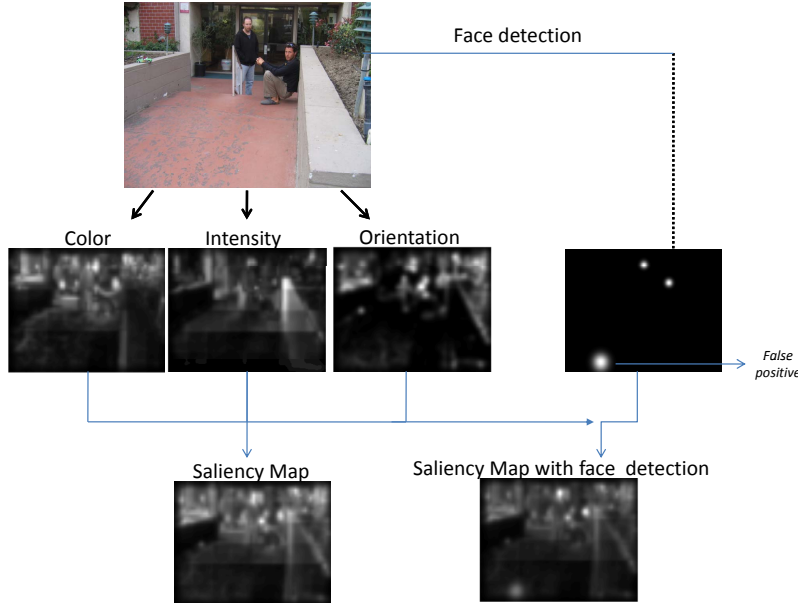

Figure 2: Modified saliency model. An image is processed through standard [5] color, orientation and intensity multi-scale channels, as well as through a trained template-matching face detection mechanism. Face coordinates and radius from the face detector are used to form a face conspicuity map (F), with peaks at facial centers. All four maps are normalized to the same dynamic range, and added with equal weights to a final saliency map (SM+VJ, or GBSM+VJ). This is compared to a saliency map which only uses the three bottom-up features maps (SM or GBSM).

## 3   Results

### 3.1   Psychophysical results

To evaluate the results of the 7 subjects' viewing of the images, we manually defined minimally sized rectangular regions-of-interest (ROIs) around each face in the entire image collection. We first assessed, in the "free-viewing" phase, how many of the first fixations went to a face, how many of the second, third fixations and so forth. In 972 out of the 1050 (7 subjects x 150 images with faces) trials (92.6%), the subject fixated on a face at least once. In 645/1050 (61.4%) trials, a

face was fixated on within the first fixation, and of the remaining 405 trials, a face was fixated on in the second fixation in 71.1% (288/405), *i.e.* after two fixations a face was fixated on in 88.9% (933/1050) of trials (Fig. 3). Given that the face ROIs were chosen very conservatively (*i.e.* fixations just next to a face do not count as fixations on the face), this shows that faces, if present, are typically fixated on within the first two fixations ($327\ ms \pm 95\ ms$ on average). Furthermore, in addition to finding early fixations on faces, we found that inter-subject scanpath consistency on images with faces was higher. For the free-viewing task, the mean minimum distance to another's subject's fixation (averaged over fixations and subjects) was 29.47 pixels on images with faces, and a greater 34.24 pixels on images without faces (different with $p < 10^{-6}$). We found similar results using a variety of different metrics (ROC, Earth Mover's Distance, Normalized Scanpath Saliency, etc.). To verify that the double spatial bias of photographer and observer ([29] for discussion of this issue) did not artificially result in high fractions of early fixations on faces, we compared our results to an unbiased baseline: for each subject, the fraction of fixations from all images which fell in the ROIs of one particular image. The null hypothesis that we would see the same fraction of first fixations on a face at random is rejected at $p < 10^{-20}$ (t-test).

To test for the hypothesis that face saliency is not due to top-down preference for faces in the absence of other interesting things, we examined the results of the "search" task, in which subjects were presented with a non-face target probe in 50% of the trials. Provided the short amount of time for the search (2 s), subjects should have attempted to tune their internal saliency weights to adjust color, intensity, and orientation optimally for the searched target [30]. Nevertheless, subjects still tended to fixate on the faces early. A face was fixated on within the first fixation in 24% of trials, within the first two fixations in 52% of trials, and within the three fixations in 77% of the trials. While this is weaker than in free-viewing, where 88.9% was achieved after just two fixations, the difference from what would be expected for random fixation selection (unbiased baseline as above) is still highly significant ($p < 10^{-8}$).

Overall, we found that in both experimental conditions ("free-viewing" and "search"), faces were powerful attractors of attention, accounting for a strong majority of early fixations when present. This trend allowed us to easily improve standard saliency models, as discussed below.

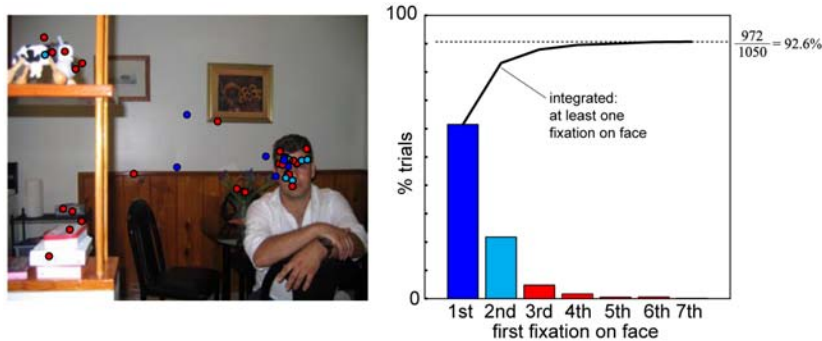

Figure 3: Extent of fixation on face regions-of-interest (ROIs) during the "free-viewing" phase . *Left:* image with all fixations (7 subjects) superimposed. First fixation marked in blue, second in cyan, remaining fixations in red. *Right:* Bars depict percentage of trials, which reach a face *the first time* in the first, second, third, ... fixation. The solid curve depicts the integral, *i.e.* the fraction of trials in which faces were fixated on at least once up to and including the $n^{th}$ fixation.

## 3.2 Assessing the saliency map models

We ran VJ on each of the 200 images used in the free viewing task, and found at least one face detection on 176 of these images, 148 of which actually contained faces (only two images with faces were missed). For each of these 176 images, we computed four saliency maps (SM, GBSM, SM+VJ, GBSM+VJ) as discussed above, and quantified the compatibility of each with our scan-path recordings, in particular fixations, using the area under an ROC curve. The ROC curves were generated by sweeping over saliency value thresholds, and treating the fraction of non-fixated pixels

on a map above threshold as false alarms, and the fraction of fixated pixels above threshold as hits [29, 31]. According to this ROC fixation "prediction" metric, for the example image in Fig. 4, all models predict above chance (50%): SM performs worst, and GBSM+VJ best, since including the face detector substantially improves performance in both cases.

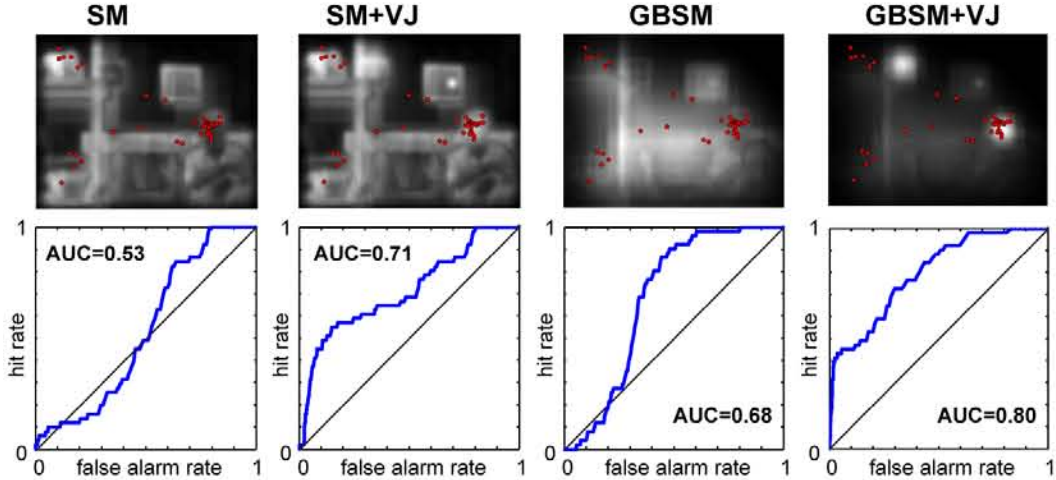

Figure 4: Comparison of the area-under-the-curve (AUC) for an image (chosen arbitrarily. Subjects' scanpaths shown on the left panels of figure 1). Top panel: image with the 49 fixations of the 7 subjects (red). First central fixations for each subject were excluded. From left to right, saliency map model of Itti et al. (SM), saliency map with the VJ face detection map (SM+VJ), the graph-based saliency map (GBSM), and the graph-based saliency map with face detection channel (GBSM+VJ). Red dots correspond to fixations. Lower panels depict ROC curves corresponding to each map. Here, GBSM+VJ predicts fixations best, as quantified by the highest AUC.

Across all 176 images, this trend prevails (Fig. 5): first, all models perform better than chance, even over the 28 images without faces. The SM+VJ model performed better than the SM model for 154/176 images. The null hypothesis to get this result by chance can be rejected at $p < 10^{-22}$ (using a coin-toss sign-test for which model does better, with uniform null-hypothesis, neglecting the size of effects). Similarly, the GBSM+VJ model performed better than the GBSM model for 142/176 images, a comparably vast majority ($p < 10^{-15}$) (see Fig. 5, right). For the 148/176 images with faces, SM+VJ was better than SM alone for 144/148 images ($p < 10^{-29}$), whereas VJ alone (equal to the face conspicuity map) was better than SM alone for 83/148 images, a fraction that fails to reach significance. Thus, although the face conspicuity map was surprisingly predictive on its own, fixation predictions were much better when it was combined with the full saliency model. For the 28 images without faces, SM (better than SM+VJ for 18) and SM+VJ (better than SM for 10) did not show a significant difference, nor did GBSM vs. GBSM+VJ (better on 15/28 compared to 13/28, respectively. However, in a recent follow-up study with more non-face images, we found preliminary results indicating that the mean ROC score of VJ-enhanced saliency maps is higher on such non-face images, although the median is slightly lower, *i.e.* performance is much improved when improved at all indicating that VJ false positives can sometimes enhance saliency maps.

In summary, we found that adding a face detector channel improves fixation prediction in images with faces dramatically, while it does not impair prediction in images without faces, even though the face detector has false alarms in those cases.

## 4 Discussion

First, we demonstrated that in natural scenes containing frontal shots of people, faces were fixated on within the first few fixations, whether subjects had to grade an image on interest value or search it for a specific *possibly non-face* target. This powerful trend motivated the introduction of a new saliency

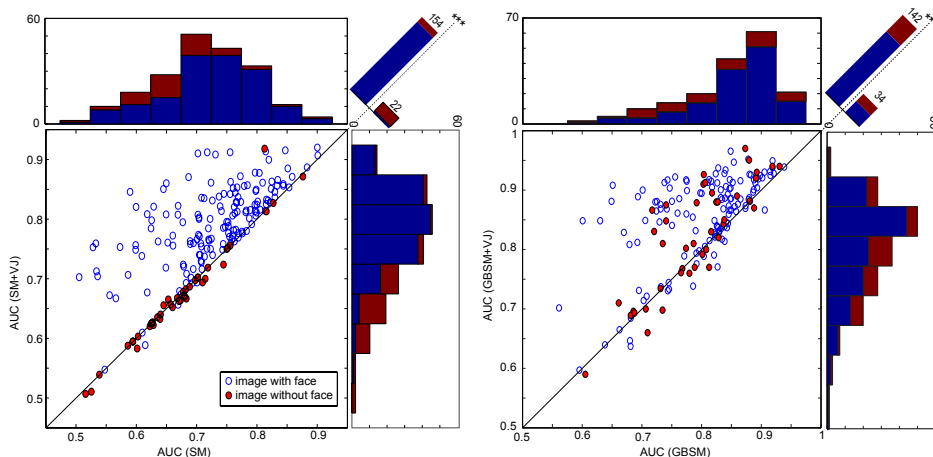

Figure 5: Performance of SM compared to SM+VJ and GBSM compared to GBSM+VJ. Scatterplots depict the area under ROC curves (AUC) for the 176 images in which VJ found a face. Each point represents a single image. Points above the diagonal indicate better prediction of the model including face detection compared to the models without face channel. Blue markers denote images with faces; red markers images without faces (*i.e.* false positives of the VJ face detector). Histograms of the SM and SM+VJ (GBSM and GBSM+VJ) are depicted to the top and left (binning: 0.05); colorcode as in scatterplots.

model, which combined the "bottom-up" feature channels of color, orientation, and intensity, with a special face-detection channel, based on the Viola & Jones algorithm. The combination was linear in nature with uniform weight distribution for maximum simplicity. In attempting to predict the fixations of human subjects, we found that this additional face channel improved the performance of both a standard and a more recent graph-based saliency model (almost all blue points in Fig. 5 are above the diagonal) in images with faces. In the few images without faces, we found that the false positives represented in the face-detection channel did not significantly alter the performance of the saliency maps – although in a preliminary follow-up on a larger image pool we found that they boost mean performance. Together, these findings point towards a specialized "face channel" in our vision system, which is subject to current debate in the attention literature [11, 12, 32].

In conclusion, inspired by biological understanding of human attentional allocation to meaningful objects - faces - we presented a new model for computing an improved saliency map which is more consistent with gaze deployment in natural images containing faces than previously studied models, even though the face detector was trained on standard sets. This suggests that faces always attract attention and gaze, relatively independent of the task. They should therefore be considered as part of the bottom-up saliency pathway.

## References

[1] G. Rizzolatti, L. Riggio, I. Dascola, and C. Umilta. Reorienting attention across the horizontal and vertical meridians: evidence in favor of a premotor theory of attention. *Neuropsychologia*, 25(1A):31–40, 1987.

[2] G.T. Buswell. *How People Look at Pictures: A Study of the Psychology of Perception in Art*. The University of Chicago press, 1935.

[3] M. Cerf, D. R. Cleary, R. J. Peters, and C. Koch. Observers are consistent when rating image conspicuity. *Vis Res*, 47(25):3017–3027, 2007.

[4] S.J. Dickinson, H.I. Christensen, J. Tsotsos, and G. Olofsson. Active object recognition integrating attention and viewpoint control. *Computer Vision and Image Understanding*, 67(3):239–260, 1997.

[5] L. Itti, C. Koch, E. Niebur, et al. A model of saliency-based visual attention for rapid scene analysis. *IEEE Transactions on Pattern Analysis and Machine Intelligence*, 20(11):1254–1259, 1998.

[6] A.L. Yarbus. *Eye Movements and Vision*. Plenum Press New York, 1967.

[7] J.M. Henderson, J.R. Brockmole, M.S. Castelhano, and M. Mack. Visual Saliency Does Not Account for Eye Movements during Visual Search in Real-World Scenes. *Eye Movement Research: Insights into Mind and Brain, R. van Gompel, M. Fischer, W. Murray, and R. Hill, Eds.*, 1997.

[8] Gregory Zelinsky, Wei Zhang, Bing Yu, Xin Chen, and Dimitris Samaras. The role of top-down and bottom-up processes in guiding eye movements during visual search. In Y. Weiss, B. Schölkopf, and J. Platt, editors, *Advances in Neural Information Processing Systems 18*, pages 1569–1576. MIT Press, Cambridge, MA, 2006.

[9] A. Torralba, A. Oliva, M.S. Castelhano, and J.M. Henderson. Contextual guidance of eye movements and attention in real-world scenes: the role of global features in object search. *Psych Rev*, 113(4):766–786, 2006.

[10] W. Einhäuser and P. König. Does luminance-contrast contribute to a saliency map for overt visual attention? *Eur. J Neurosci*, 17(5):1089–1097, 2003.

[11] O. Hershler and S. Hochstein. At first sight: a high-level pop out effect for faces. *Vision Res*, 45(13):1707–24, 2005.

[12] R. Vanrullen. On second glance: Still no high-level pop-out effect for faces. *Vision Res*, 46(18):3017–3027, 2006.

[13] C. Simion and S. Shimojo. Early interactions between orienting, visual sampling and decision making in facial preference. *Vision Res*, 46(20):3331–3335, 2006.

[14] R. Adolphs. Neural systems for recognizing emotion. *Curr. Op. Neurobiol.*, 12(2):169–177, 2002.

[15] A. Klin, W. Jones, R. Schultz, F. Volkmar, and D. Cohen. Visual Fixation Patterns During Viewing of Naturalistic Social Situations as Predictors of Social Competence in Individuals With Autism, 2002.

[16] JJ Barton. Disorders of face perception and recognition. *Neurol Clin*, 21(2):521–48, 2003.

[17] K.K. Sung and T. Poggio. Example-based learning for view-based human face detection. *IEEE Transactions on Pattern Analysis and Machine Intelligence*, 20(1):39–51, 1998.

[18] H.A. Rowley, S. Baluja, and T. Kanade. Neural network-based face detection. *IEEE Transactions on Pattern Analysis and Machine Intelligence*, 20(1):23–38, 1998.

[19] H. Schneiderman and T. Kanade. Statistical method for 3 D object detection applied to faces and cars. *Computer Vision and Pattern Recognition*, 1:746–751, 2000.

[20] D. Roth, M. Yang, and N. Ahuja. A snow-based face detection. In S. A. Solla, T. K. Leen, and K. R. Muller, editors, *Advances in Neural Information Processing Systems 13*, pages 855–861. MIT Press, Cambridge, MA, 2000.

[21] P. Viola and M. Jones. Rapid object detection using a boosted cascade of simple features. *Computer Vision and Pattern Recognition*, 1:511–518, 2001.

[22] D. Walther. *Interactions of visual attention and object recognition: computational modeling, algorithms, and psychophysics*. PhD thesis, California Institute of Technology, 2006.

[23] C. Breazeal and B. Scassellati. A context-dependent attention system for a social robot. *1999 International Joint Conference on Artificial Intelligence*, pages 1254–1259, 1999.

[24] V. Navalpakkam and L. Itti. Search Goal Tunes Visual Features Optimally. *Neuron*, 53(4):605–617, 2007.

[25] D.H. Brainard. The psychophysics toolbox. *Spat Vis*, 10(4):433–436, 1997.

[26] F.W. Cornelissen, E.M. Peters, and J. Palmer. The Eyelink Toolbox: Eye tracking with MATLAB and the Psychophysics Toolbox. *Behav Res Meth Instr Comput*, 34(4):613–617, 2002.

[27] J. Harel, C. Koch, and P. Perona. Graph-based visual saliency. In B. Schölkopf, J. Platt, and T. Hoffman, editors, *Advances in Neural Information Processing Systems 19*, pages 545–552. MIT Press, Cambridge, MA, 2007.

[28] G. Bradski, A. Kaehler, and V. Pisarevsky. Learning-based computer vision with Intels open source computer vision library. *Intel Technology Journal*, 9(1), 2005.

[29] B.W. Tatler, R.J. Baddeley, and I.D. Gilchrist. Visual correlates of fixation selection: effects of scale and time. *Vision Res*, 45(5):643–59, 2005.

[30] V. Navalpakkam and L. Itti. Search goal tunes visual features optimally. *Neuron*, 53(4):605–617, 2007.

[31] R.J. Peters, A. Iyer, L. Itti, and C. Koch. Components of bottom-up gaze allocation in natural images. *Vision Res*, 45(18):2397–2416, 2005.

[32] O. Hershler and S. Hochstein. With a careful look: Still no low-level confound to face pop-out Authors' reply. *Vis Res*, 46(18):3028–3035, 2006.

